# Central and Pairwise Data Clustering by Competitive Neural Networks

**Joachim Buhmann** & **Thomas Hofmann**
Rheinische Friedrich–Wilhelms–Universität
Institut für Informatik II, Römerstraße 164
D-53117 Bonn, Fed. Rep. Germany

## Abstract

Data clustering amounts to a combinatorial optimization problem to reduce the complexity of a data representation and to increase its precision. Central and pairwise data clustering are studied in the maximum entropy framework. For central clustering we derive a set of reestimation equations and a minimization procedure which yields an optimal number of clusters, their centers and their cluster probabilities. A meanfield approximation for pairwise clustering is used to estimate assignment probabilities. A selfconsistent solution to multidimensional scaling and pairwise clustering is derived which yields an optimal embedding and clustering of data points in a $d$-dimensional Euclidian space.

## 1  Introduction

A central problem in information processing is the reduction of the data complexity with minimal loss in precision to discard noise and to reveal basic structure of data sets. Data clustering addresses this tradeoff by optimizing a cost function which preserves the original data as complete as possible and which simultaneously favors prototypes with minimal complexity (Linde *et al.*, 1980; Gray, 1984; Chou *et al.*, 1989; Rose *et al.*, 1990). We discuss an objective function for the *joint optimization* of distortion errors and the complexity of a reduced data representation. A maximum entropy estimation of the cluster assignments yields a unifying framework for clustering algorithms with a number of different distortion and complexity measures. The close analogy of complexity optimized clustering with *winner-take-all* neural networks suggests a neural-like implementation resembling topological feature maps (see Fig. 1).

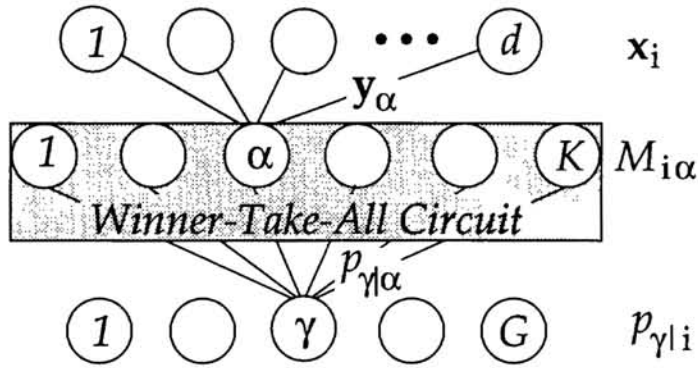

Figure 1: Architecture of a three layer competitive neural network for central data clustering with $d$ neurons in the input layer, $K$ neurons in the clustering layer with activity $\langle M_{i\alpha} \rangle$ and $G$ neurons in the classification layer. The output neurons estimate the conditional probability $p_{\gamma|i}$ of data point $i$ being in class $\gamma$.

Given is a set of data points which are characterized either by coordinates $\{\mathbf{x}_i | \mathbf{x}_i \in \Re^d; \ i = 1, \ldots, N\}$ or by pairwise distances $\{\mathcal{D}_{ik} | i, k = 1, \ldots, N\}$. The goal of data clustering is to determine a partitioning of a data set which either minimizes the average distance of data points to their cluster centers or the average distance between data points of the same cluster. The two cases are refered to as central or pairwise clustering. Solutions to central clustering are represented by a set of data prototypes $\{\mathbf{y}_\alpha | \mathbf{y}_\alpha \in \Re^d; \ \alpha = 1, \ldots, K\}$, and the size $K$ of that set. The assignments $\{M_{i\alpha} | \alpha = 1, \ldots, K; i = 1, \ldots, N\}$, $M_{i\alpha} \in \{0, 1\}$ denote that data point $i$ is uniquely assigned to cluster $\alpha$ ($\sum_\nu M_{i\nu} = 1$). Rate distortion theory specifies the optimal choice of $\mathbf{y}_\alpha$ being the cluster centroids, i.e., $\sum_i M_{i\alpha} \frac{\partial}{\partial \mathbf{y}_\alpha} \mathcal{D}_{i\alpha}(\mathbf{x}_i, \mathbf{y}_\alpha) = 0$. Given only a set of distances or dissimilarities the solution to pairwise clustering is characterized by the expected assignment variables $\langle M_{i\alpha} \rangle$. The complexity $\{\mathcal{C}_\alpha | \alpha = 1, \ldots, K\}$ of a clustering solution depends on the specific information processing application at hand, in particular, we assume that $\mathcal{C}_\alpha$ is only a function of the cluster probability $p_\alpha = \sum_{i=1}^N M_{i\alpha}/N$. We propose the central clustering cost function

$$\mathcal{E}_K^{\text{cc}}(\{M_{i\nu}\}) = \sum_{i=1}^N \sum_{\nu=1}^K M_{i\nu} \Big( \mathcal{D}_{i\nu}(\mathbf{x}_i, \mathbf{y}_\nu) + \lambda \mathcal{C}_\nu(p_\nu) \Big) \tag{1}$$

and the pairwise clustering cost function

$$\mathcal{E}_K^{\text{pc}}(\{M_{i\nu}\}) = \sum_{i=1}^N \sum_{\nu=1}^K M_{i\nu} \Big( \frac{1}{2p_\nu N} \sum_{k=1}^N M_{k\nu} \mathcal{D}_{ik} + \lambda \mathcal{C}_\nu(p_\nu) \Big). \tag{2}$$

The distortion and complexity costs are adjusted in size by the weighting parameter $\lambda$. The cost functions (1,2) have to be optimized in an iterative fashion: (i) vary the assignment variables $M_{i\alpha}$ for a fixed number $K$ of clusters such that the costs $\mathcal{E}_K^{\text{cc,pc}}(\{M_{i\alpha}\})$ decrease; (ii) increment the number of clusters $K \to K + 1$ and optimize $M_{i\alpha}$ again.

Complexity costs which penalize small, sparsely populated clusters, i.e., $\mathcal{C}_\alpha = 1/p_\alpha^s, s = 1, 2, \ldots$, favor equal cluster probabilities, thereby emphasizing the hardware aspect of a clustering solution. The special case $s = 1$ with constant costs per cluster corresponds to $K$-means clustering. An alternative complexity measure which estimates encoding costs for data compression and data transmission is the Shannon entropy of a cluster set $\langle \mathcal{C} \rangle \equiv \sum_\nu p_\nu \mathcal{C}_\nu = - \sum_\nu p_\nu \log p_\nu$.

The most common choice for the distortion measure are distances $\mathcal{D}_{i\alpha} = \|\mathbf{x}_i - \mathbf{y}_\alpha\|^r$ which preserve the permutation symmetry of (1) with respect to the cluster index $\nu$. A data partitioning scheme without permutation invariance of cluster indices is described by the cost function

$$\mathcal{E}_K^{\mathrm{T}} = \sum_i \sum_\nu M_{i\nu}\Big(\langle\!\langle \mathcal{D}_{i\nu} \rangle\!\rangle + \lambda \mathcal{C}(p_\nu)\Big). \tag{3}$$

The generalized distortion error $\langle\!\langle \mathcal{D}_{i\alpha} \rangle\!\rangle \equiv \sum_\gamma \mathrm{T}_{\alpha\gamma} \mathcal{D}_{i\gamma}(\mathbf{x}_i, \mathbf{y}_\gamma)$ between data point $\mathbf{x}_i$ and cluster center $\mathbf{y}_\alpha$ quantifies the intrinsic quantization errors $\mathcal{D}_{i\gamma}(\mathbf{x}_i, \mathbf{y}_\gamma)$ and the additional errors due to transitions $\mathrm{T}_{\alpha\gamma}$ from index $\gamma$ to $\alpha$. Such transitions might be caused by noise in communication channels. These index transitions impose a topological order on the set of indices $\{\alpha | \alpha = 1, \ldots, K\}$ which establishes a connection to self-organizing feature maps (Kohonen, 1984; Ritter *et al.*, 1992) in the case of nearest neighbor transitions in a $d$-dimensional index space. We refer to such a partitioning of the data space as topology preserving clustering.

## 2   Maximum Entropy Estimation of Central Clustering

Different combinations of complexity terms, distortion measures and topology constraints define a variety of central clustering algorithms which are relevant in very different information processing contexts. To derive robust, preferably parallel algorithms for these data clustering cases, we study the clustering optimization problem in the probabilistic framework of maximum entropy estimation. The resulting Gibbs distribution proved to be the most stable distribution with respect to changes in expected clustering costs (Tikochinsky *et al.*, 1984) and, therefore, has to be considered *optimal* in the sense of robust statistics. Statistical physics (see e.g. (Amit, 1989; Rose *et al.*, 1990)) states that maximizing the entropy at a fixed temperature $T = 1/\beta$ is equivalent to minimizing the free energy

$$
\begin{aligned}
\mathcal{F}_K &= -T \ln \mathcal{Z} = -T \ln\Big( \sum_{\{M_{i\nu}\}} \exp(-\beta \mathcal{E}_K) \Big) \\
&= -\lambda N \sum_\nu p_\nu{}^2 \frac{\partial \mathcal{C}_\nu}{\partial p_\nu} - \frac{1}{\beta} \sum_i \log\Big( \sum_\nu \exp\big[-\beta(\langle\!\langle \mathcal{D}_{i\nu} \rangle\!\rangle + \lambda \mathcal{C}_\nu^*)\big] \Big) \tag{4}
\end{aligned}
$$

with respect to the variables $p_\nu, \mathbf{y}_\nu$. The effective complexity costs are $\mathcal{C}_\nu^* \equiv \partial (p_\nu \mathcal{C}_\nu)/\partial p_\nu$. For a derivation of (4) see (Buhmann, Kühnel, 1993b).

The resulting re-estimation equations for the expected cluster probabilities and the expected centroid positions are necessary conditions of $\mathcal{F}_K$ being minimal, i.e.

$$p_\alpha = \frac{1}{N} \sum_i \langle M_{i\alpha} \rangle, \tag{5}$$

$$0 = \frac{1}{N} \sum_i \sum_\gamma \mathrm{T}_{\gamma\alpha} \langle M_{i\gamma} \rangle \frac{\partial}{\partial \mathbf{y}_\alpha} \mathcal{D}_{i\alpha}(\mathbf{x}_i, \mathbf{y}_\alpha), \tag{6}$$

$$\langle M_{i\alpha} \rangle = \frac{\exp\big[-\beta(\langle\!\langle \mathcal{D}_{i\alpha} \rangle\!\rangle + \lambda \mathcal{C}_\alpha^*)\big]}{\displaystyle\sum_{\nu=1}^K \exp[-\beta(\langle\!\langle \mathcal{D}_{i\nu} \rangle\!\rangle + \lambda \mathcal{C}_\nu^*)]}. \tag{7}$$

The expectation value $\langle M_{i\alpha} \rangle$ of the assignment variable $M_{i\alpha}$ can be interpreted as a fuzzy membership of data point $\mathbf{x}_i$ in cluster $\alpha$. The case of supervised clustering can be treated in an analogous fashion (Buhmann, Kühnel, 1993a) which gives rise to the third layer in the neural network implementation (see Fig. 1). The global minimum of the free energy (4) with respect to $p_\alpha, \mathbf{y}_\alpha$ determines the maximum entropy solution of the cost function (1). Note that the optimization problem (1) of a $K^N$ state space has been reduced to a $K(d+1)$ dimensional minimization of the free energy $\mathcal{F}_K$ (4). To find the optimal parameters $p_\alpha, \mathbf{y}_\alpha$ and the number of clusters $K$ which minimize the free energy, we start with one cluster located at the centroid of the data distribution, split that cluster and reestimate $p_\alpha, \mathbf{y}_\alpha$ using equation (5,6). The new configuration is accepted as an improved solution if the free energy (4) has been decreased. This splitting and reestimation loop is continued until we fail to find a new configuration with lower free energy. The temperature determines the fuzziness of a clustering solution, whereas the complexity term penalizes excessively many clusters.

## 3 Meanfield Approximation for Pairwise Clustering

The maximum entropy estimation for pairwise clustering constitutes a much harder problem than the calculation of the free energy for central clustering. Analytical expression for the Gibbs distributions are not known except for the quadratic distance measure $\mathcal{D}_{ik} = (\mathbf{x}_i - \mathbf{x}_k)^2$. Therefore, we approximate the free energy by a variational principle commonly refered to as meanfield approximation. Given the costfunction (2) we derive a lower bound to the free energy by a system of noninteracting assignment variables. The approximative costfunction with the variational parameters $\mathcal{E}_{i\nu}$ is

$$\mathcal{E}_K^0 = \sum_{\nu=1}^{K} \sum_{i=1}^{N} M_{i\nu} \mathcal{E}_{i\nu}. \tag{8}$$

The original costfunction for pairwise clustering can be written as $\mathcal{E}_K^{pc} = \mathcal{E}_K^0 + V$ with a (small) perturbation term $V = \mathcal{E}_K^{pc} - \mathcal{E}_K^0$ due to cluster interactions. The partition function

$$
\begin{aligned}
\mathcal{Z} &\equiv \sum_{\{M_{i\nu}\}} \exp\left(-\beta \mathcal{E}_K^{pc}\right) = \sum_{\{M_{i\nu}\}} \exp\left(-\beta \mathcal{E}_K^0\right) \frac{\displaystyle\sum_{\{M_{i\nu}\}} \exp\left(-\beta \mathcal{E}_K^0\right) \exp\left(-\beta V\right)}{\displaystyle\sum_{\{M_{i\nu}\}} \exp\left(-\beta \mathcal{E}_K^0\right)} \\
&= \mathcal{Z}_0 \langle \exp(-\beta V) \rangle_0 > \mathcal{Z}_0 \exp(-\beta \langle V \rangle_0)
\end{aligned}
\tag{9}
$$

is bound from below if terms of the order $\mathcal{O}(\langle (V - \langle V \rangle_0)^3 \rangle_0)$ and higher are negligible compared to the quadratic term. The angular brackets denote averages over all configurations of the costfunction without interactions. The averaged perturbation term $\langle V \rangle_0$ amounts to

$$\langle V \rangle_0 = \sum_{\nu} \sum_{i,k} \langle M_{i\nu} \rangle \langle M_{k\nu} \rangle \frac{1}{2p_\nu N} \mathcal{D}_{ik} + \lambda \sum_{\nu} \sum_{i} \langle M_{i\nu} \rangle \mathcal{C}_\nu - \sum_{\nu} \sum_{i} \langle M_{i\nu} \rangle \mathcal{E}_{i\nu}. \tag{10}$$

$\langle M_{i\alpha} \rangle$ being the averaged assignment variables

$$\langle M_{i\alpha} \rangle = \frac{\exp(-\beta \mathcal{E}_{i\alpha})}{\displaystyle\sum_{\nu} \exp(-\beta \mathcal{E}_{i\nu})}. \tag{11}$$

The meanfield approximation with the cost function (8) yields a lower bound to the partition function $\mathcal{Z}$ of the original pairwise clustering problem. Therefore, we vary the parameters $\mathcal{E}_{i\alpha}$ to maximize the quantity $\ln \mathcal{Z}_0 - \beta \langle V \rangle_0$ which produces the best lower bound of $\mathcal{Z}$ based on an interaction free costfunction. Variation of $\mathcal{E}_{i\alpha}$ leads to the conditions

$$\sum_{\nu} \frac{\partial \langle M_{i\nu} \rangle}{\partial \mathcal{E}_{i\alpha}} \left( \mathcal{E}_{i\nu} - \mathcal{E}_{i\nu}^* \right) = 0, \quad \forall i \in \{1, ..., N\}, \alpha \in \{1, ..., K\}, \tag{12}$$

$\mathcal{E}_{i\nu}^*$ being defined as

$$\mathcal{E}_{i\nu}^* = \frac{1}{p_\nu N} \sum_j \langle M_{j\nu} \rangle \left( \mathcal{D}_{ij} - \frac{1}{2 p_\nu N} \sum_k \langle M_{k\nu} \rangle \mathcal{D}_{jk} \right) + \lambda C_\nu^*. \tag{13}$$

For a given distance matrix $\mathcal{D}_{ik}$ the transcendental equations (11,12) have to be solved simultaneously.

So far the $\mathcal{E}_{i\alpha}$ have been treated as independent variation parameters. An important problem, which is usually discussed in the context of *Multidimensional Scaling*, is to find an embedding for the data set in an Euclidian space and to cluster the embedded data. The variational framework can be applied to this problem, if we consider the parameters $\mathcal{E}_{i\alpha}$ as functions of data coordinates and prototype coordinates, $\mathcal{E}_{i\alpha} = \mathcal{D}_{i\alpha}(\mathbf{x}_i, \mathbf{y}_\alpha)$, e.g. with a quadratic distortion measure $\mathcal{D}_{i\alpha}(\mathbf{x}_i, \mathbf{y}_\alpha) = \|\mathbf{x}_i - \mathbf{y}_\alpha\|^2$. The variables $\mathbf{x}_i, \mathbf{y}_\alpha \in \Re^d$ are the variational parameters which have to be determined by maximizing $\ln \mathcal{Z}_0 - \beta \langle V \rangle_0$. Without imposing the restriction for the prototypes to be the cluster centroids, this leads to the following conditions for the data coordinates

$$\sum_{\nu} \langle M_{i\nu} \rangle \left( \mathcal{E}_{i\nu} - \mathcal{E}_{i\nu}^* \right) \left( \mathbf{y}_\nu - \sum_\mu \langle M_{i\mu} \rangle \mathbf{y}_\mu \right) = 0, \quad \forall i \in \{1, ..., N\}. \tag{14}$$

After further algebraic manipulations we receive the explicit expression for the data points

$$\mathcal{K}_i \mathbf{x}_i = \frac{1}{2} \sum_\nu \langle M_{i\nu} \rangle \left( \|\mathbf{y}_\nu\|^2 - \mathcal{E}_{i\nu}^* \right) \left( \mathbf{y}_\nu - \sum_\mu \langle M_{i\mu} \rangle \mathbf{y}_\mu \right), \tag{15}$$

with the covariance matrix $\mathcal{K}_i = \left( \langle \mathbf{y} \mathbf{y}^T \rangle_i - \langle \mathbf{y} \rangle_i \langle \mathbf{y} \rangle_i^T \right)$, $\langle \mathbf{y} \rangle_i = \sum_\nu \langle M_{i\nu} \rangle \mathbf{y}_\nu$. Let us assume that the matrix $\mathcal{K}_i$ is non–singular which imposes the condition $K > d$ and the cluster centers $\{\mathbf{y}_\alpha | \alpha = 1, \ldots, K\}$ being in general position. For $K \le d$ the equations $\mathcal{E}_{i\alpha} = \mathcal{E}_{i\alpha}^* + c_i$ are exactly solvable and embedding in dimensions larger than $K$ produces non–unique solutions without improving the lower bound in (9).

Varying $\ln \mathcal{Z}_0 - \beta \langle V \rangle_0$ with respect to $\mathbf{y}_\alpha$ yields a second set of stationarity conditions

$$\sum_j \langle M_{j\alpha} \rangle \left( 1 - \langle M_{j\alpha} \rangle \right) \left( \mathcal{E}_{j\alpha} - \mathcal{E}_{j\alpha}^* \right) \left( \mathbf{x}_j - \mathbf{y}_\alpha \right) = 0, \quad \forall \alpha \in \{1, ..., K\}. \tag{16}$$

The weighting factors in (16), however, decay exponentially fast with the inverse temperature, i.e., $\langle M_{j\alpha} \rangle \left( 1 - \langle M_{j\alpha} \rangle \right) \sim \mathcal{O} \left( \beta \exp[-\beta c] \right)$, $c > 0$. This implies that the optimal solution for the data coordinates displays only a very weak dependence on the special choice of the prototypes in the low temperature regime. Fixing the parameters $\mathbf{y}_\alpha$ and solving the transcendental equations (14,15) for $\mathbf{x}_i$, the solution will be very close to the optimal approximation. It is thus possible to choose the prototypes as the cluster centroids $\mathbf{y}_\alpha = 1/(p_\alpha N) \sum_i \langle M_{i\alpha} \rangle \mathbf{x}_i$ and, thereby, to solve Eq. (15) in a self–consistent fashion.

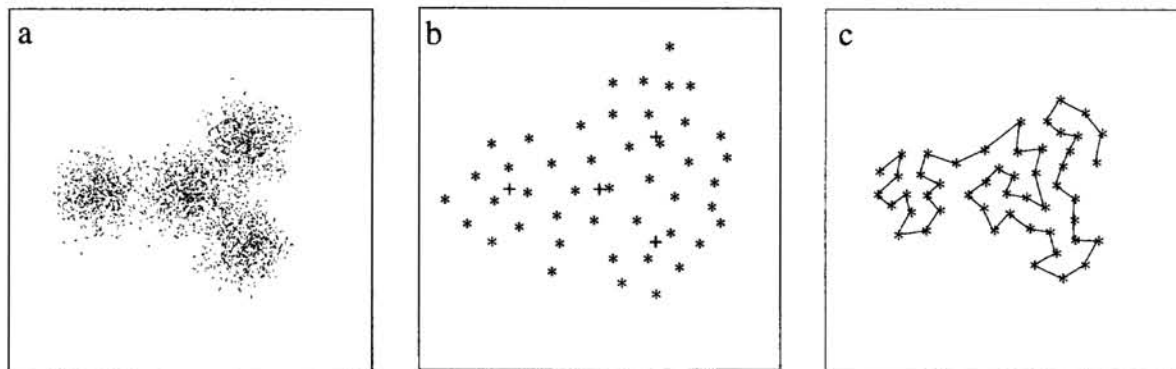

Figure 2: A data distribution (4000 data points) (a), generated by four normally distributed sources is clustered with the complexity measure $C_\alpha = -\log p_\alpha$ and $\lambda = 0.4$ (b). The plus signs (+) denote the centers of the Gaussians and stars ($\star$) denote cluster centers. Figure (c) shows a topology preserving clustering solution with complexity $C_\alpha = 1/p_\alpha$ and external noise ($\eta = 0.05$).

If the prototype variables depend on the data coordinates, the derivatives $\partial \mathbf{y}_\alpha / \partial \mathbf{x}_i$ will not vanish in general and the condition (14) becomes more complicated. Regardless of this complication the resulting algorithm to estimate data coordinates $\mathbf{x}_i$ interleaves the clustering process and the optimization of the embedding in a Euclidian space. The artificial separation of multidimensional scaling from data clustering has been avoided. Data points are embedded and clustered simultaneously. Furthermore, we have derived a maximum entropy approximation which is most robust with respect to changes in the average costs $\langle \mathcal{E}_K \rangle$.

## 4    Clustering Results

Non-topological ($T_{\alpha\gamma} = \delta_{\alpha\gamma}$) clustering results at zero temperature for the logarithmic complexity measure ($C_\alpha = \log p_\alpha$) are shown in Fig. 2b. In the limit of very small complexity costs the best clustering solution densely covers the data distribution. The specific choice of logarithmic complexity costs causes an almost homogeneous density of cluster centers, a phenomenon which is known from studies of asymptotic codebook densities and which is explained by the vanishing average complexity costs $\langle C_\alpha \rangle = -p_\alpha \log p_\alpha$ of very sparsely occupied clusters (for references see (Buhmann, Kühnel, 1993b)).

Figure 2c shows a clustering configuration assuming a one-dimensional topology in index space with nearest neighbor transitions. The short links between neighboring nodes of the neural chain indicate that the distortions due to cluster index transitions have also been optimized. Note, that complexity optimized clustering determines the length of the chain or, for a more general noise distribution, an optimal size of the cluster set. This stopping criterion for adding new cluster nodes generalizes self-organizing feature maps (Kohonen, 1984) and removes arbitrariness in the design of topological mappings. Furthermore, our algorithm is derived from an energy minimization principle in contrast to self-organizing feature maps which "cannot be derived as a stochastic gradient on *any* energy function" (Erwin *et al.*, 1992).

The complexity optimized clustering scheme has been tested on the real world task of

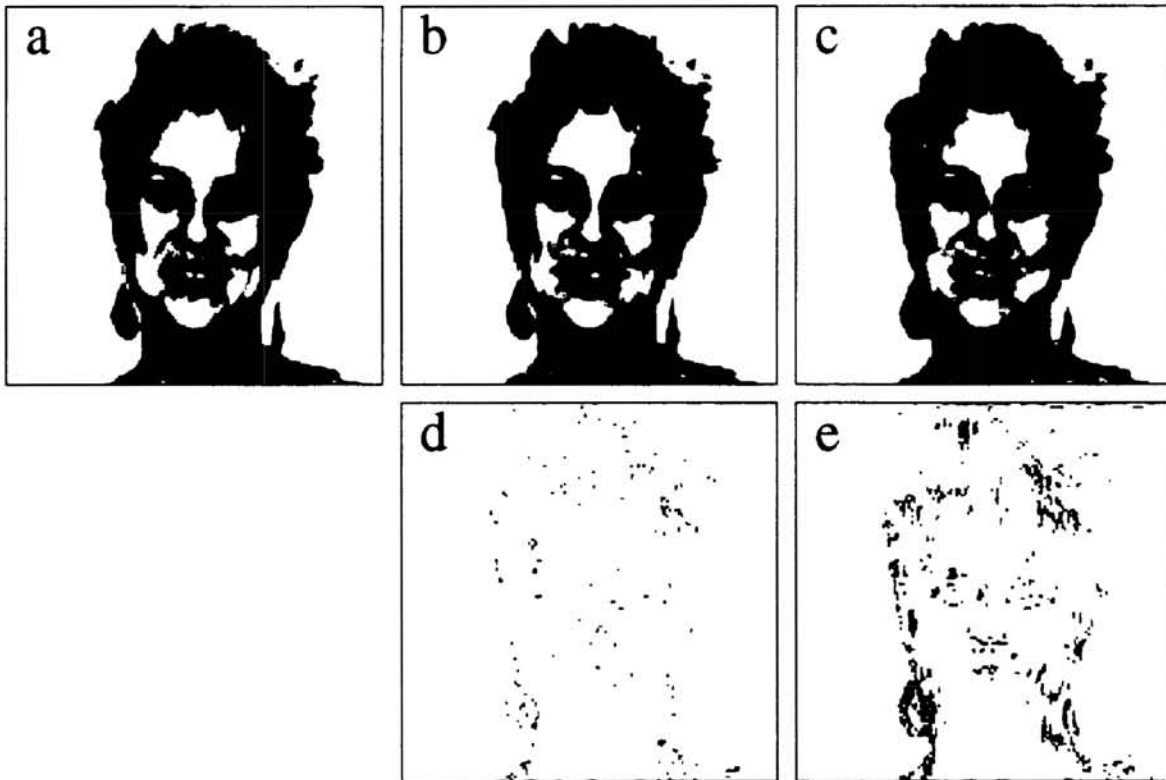

Figure 3: Quantization of a 128×128, 8bit, gray-level image. (a) Original picture. (b) Image reconstruction from wavelet coefficients quantized with entropic complexity. (c) Reconstruction from wavelet coefficients quantized by $K$-means clustering. (d,e) Absolute values of reconstruction errors in the images (b,c). Black is normalized in (d,e) to a deviation of 92 gray values.

image compression (Buhmann, Kühnel, 1993b). Entropy optimized clustering of wavelet decomposed images has reduced the reconstruction error of the compressed images up to 30 percent. Images of a compression and reconstruction experiment are shown in Fig. 3. The compression ratio is 24.5 for a 128 × 128 image. According to our efficiency criterion entropy optimized compression is 36.8% more efficient than $K$-means clustering for that compression factor. The peak SNR values for (b,c) are 30.1 and 27.1, respectively. The considerable higher error near edges in the reconstruction based on $K$-means clustering (e) demonstrates that entropy optimized clustering of wavelet coefficients not only results in higher compression ratios but, even more important it preserves psychophysically important image features like edges more faithfully than conventional compression schemes.

## 5   Conclusion

Complexity optimized clustering is a maximum entropy approach to central and pairwise data clustering which determines the optimal number of clusters as a compromise between distortion errors and the complexity of a cluster set. The complexity term turns out to be as important for the design of a cluster set as the distortion measure. Complexity optimized clustering maps onto a *winner-take-all* network which suggests hardware implementations in analog VLSI (Andreou *et al.*, 1991). Topology preserving clustering provides us with a

cost function based approach to limit the size of self-organizing maps.

The maximum entropy estimation for pairwise clustering cannot be solved analytically but has to be approximated by a meanfield approach. This meanfield approximation of the pairwise clustering costs with quadratic Euclidian distances establishes a connection between multidimensional scaling and clustering. Contrary to the usual strategy which embeds data according to their dissimilarities in a Euclidian space and, in a separate second step, clusters the embedded data, our approach finds the Euclidian embedding and the data clusters simultaneously and in a selfconsistent fashion.

The proposed framework for data clustering unifies traditional clustering techniques like $K$-means clustering, entropy constraint clustering or fuzzy clustering with neural network approaches such as topological vector quantizers. The network size and the cluster parameters are determined by a problem adapted complexity function which removes considerable arbitrariness present in other non-parametric clustering methods.

**Acknowledgement**: JB thanks H. Kühnel for insightful discussions. This work was supported by the Ministry of Science and Research of the state Nordrhein-Westfalen.

# References

Amit, D. (1989). *Modelling Brain Function*. Cambridge: Cambridge University Press.

Andreou, A. G., Boahen, K. A., Pouliquen, P. O., Pavasović, A., Jenkins, R. E., Strohbehn, K. (1991). Current Mode Subthreshold MOS Circuits for Analog VLSI Neural Systems. *IEEE Transactions on Neural Networks*, **2**, 205–213.

Buhmann, J., Kühnel, H. (1993a). Complexity Optimized Data Clustering by Competitive Neural Networks. *Neural Computation*, **5**, 75–88.

Buhmann, J., Kühnel, H. (1993b). Vector Quantization with Complexity Costs. *IEEE Transactions on Information Theory*, **39**(4), 1133–1145.

Chou, P. A., Lookabaugh, T., Gray, R. M. (1989). Entropy-Constrained Vector Quantization. *IEEE Transactions on Acoustics, Speech and Signal Processing*, **37**, 31–42.

Erwin, W., Obermayer, K., Schulten, K. (1992). Self-organizing Maps: Ordering, Convergence Properties, and Energy Functions. *Biological Cybernetics*, **67**, 47–55.

Gray, R. M. (1984). Vector Quantization. *IEEE Acoustics, Speech and Signal Processing Magazine*, April, 4–29.

Kohonen, T. (1984). *Self–organization and Associative Memory*. Berlin: Springer.

Linde, Y., Buzo, A., Gray, R. M. (1980). An algorithm for vector quantizer design. *IEEE Transactions on Communications COM*, **28**, 84–95.

Ritter, H., Martinetz, T., Schulten, K. (1992). *Neural Computation and Self-organizing Maps*. New York: Addison Wesley.

Rose, K., Gurewitz, E., Fox, G. (1990). Statistical Mechanics and Phase Transitions in Clustering. *Physical Review Letters*, **65**(8), 945–948.

Tikochinsky, Y., Tishby, N.Z., Levine, R. D. (1984). Alternative Approach to Maximum–Entropy Inference. *Physical Review A*, **30**, 2638–2644.